# Application of Neural Network Methodology to the Modelling of the Yield Strength in a Steel Rolling Plate Mill

**Ah Chung Tsoi**
Department of Electrical Engineering
University of Queensland,
St Lucia, Queensland 4072,
Australia.

## Abstract

In this paper, a tree based neural network viz. MARS (Friedman, 1991) for the modelling of the yield strength of a steel rolling plate mill is described. The inputs to the time series model are temperature, strain, strain rate, and interpass time and the output is the corresponding yield stress. It is found that the MARS-based model reveals which variable's functional dependence is nonlinear, and significant. The results are compared with those obtained by using a Kalman filter based online tuning method and other classification methods, e.g. CART, C4.5, Bayesian classification. It is found that the MARS-based method consistently outperforms the other methods.

## 1 Introduction

Hot rolling of steel slabs into flat plates is a common process in a steel mill. This technology has been in use for many years. The process of rolling hot slabs into plates is relatively well understood [see, e.g., Underwood, 1950]. But with the intense intrnational market competition, there is more and more demand on the quality of the finished plates. This demand for quality fuels the search for a better understanding of the underlying mechanisms of the transformation of hot slabs into plates, and a better control of the parameters involved. Hopefully, a better understanding of the controlling parameters will lead to a more optimal setting of the control on the process, which will lead ultimately to a better quality final product.

In this paper, we consider the problem of modelling the plate yield stress in a hot steel rolling plate mill. Rolling is a process of plastic deformation and its objective is achieved by subjecting the material to forces of such a magnitude that the resulting stresses produce permanent change of shape. Apart from the obvious dependence on the materials used, the characteristics of the material undergoing plastic deformation are described by stress, strain and temperature, if the rolling is performed on hot slabs. In addition, the interpass time, i.e., the time between passes of the slab through the rollers (an indirect measure of the rolling velocity), directly influences the metallurgical structure of the metal during rolling.

There is considerable evidence that the yield stress is also dependent on the strain rate. In fact, it is observed that as the strain rate increases, the initial yield point increases appreciably, but after an extension is achieved, the effect of strain rate on the yield stress is very much reduced [see, e.g., Underwood, 1950].

The effect of temperature on the yield stress is important. It is shown that the resistance to deformation increases with a decrease in temperature. The resistance to deformation versus temperature diagram shows a "hump" in the curve, which corresponds to the temperature at which the structure of material changes fundamentally [see, e.g., Underwood, 1950, Hodgson & Collinson, 1990].

Using, e.g., an energy method, it is possible to formulate a theoretical model of the dependence of deformation resistance on temperature, strain, strain rate, velocity (indirectly, the interpass time). One may then validate the theoretical model by performing a rolling experiment on a piece of material, perhaps under laboratory conditions [see .e.g., Horihata, Motomura, 1988, for consideration of a three roller system].

It is difficult to apply the derived theoretical model to a practical situation, due to the fact that in a practical process, the measurement of strain and strain rate are not accurate. Secondly, one cannot possibly perform a rolling experiment on each new piece of material to be rolled. Thus though the theoretical model may serve as a guide to our understanding of the process, it is not suitable for controller design purposes.

There are empirical models relating the resistance of deformation to temperature, strain and strain rate [see, e.g., Underwood, 1950, for an account of older models]. These models are often obtained by fitting the observed data to a general data model.

The following model has been found useful in fitting the observed practical data

$$k_m = a\epsilon^b \sinh^{-1}(c\dot{\epsilon}\exp(\frac{d}{T})^f) \qquad (1)$$

where $k_m$ is the yield stress, $\epsilon$ is the strain, $\dot{\epsilon}$ is the corresponding strain rate, and $T$ is the temperature. $a, b, c, d$ and $f$ are unknown constants. It is claimed that this model will give a good prediction of the yield stress, especially at lower temperatures, and for thin plate passes [Hodgson & Collinson, 1990].

This model does not always give good predictions over all temperatures as mill conditions vary with time, and the model is only "tuned" on a limited set of data.

In order to overcome this problem, McFarlane, Telford, and Petersen [1991] have experimented with a recursive model based on the Kalman filter in control theory to update the parameters (see, e.g. Anderson, Moore, [1980]), $a, b, c, d, f$ in the above model. To better describe the material behaviour at different temperatures, the model explicitly incorporates two separate sub-models with a temperature dependence:

1. Full crystallisation ($T < T_{upper}$)

$$k_m = a\epsilon^b \sinh^{-1}(c\dot{\epsilon} \exp(\frac{d}{T})^f) \tag{2}$$

The constants $a, b, c, d, f$ are model coefficients.

2. Partial recrystallisation ($T_{lower} \leq T \leq T_{upper}$).

$$k_m = a(\epsilon + \epsilon^*)^b \sinh^{-1}(c\dot{\epsilon} \exp(\frac{d}{T})^f) \tag{3}$$

$$t_{0.5} = j(\lambda_{i-1}\epsilon_{i-1} + \epsilon_i)^g f_1((q(T_{i-1}, T_i)h)) \tag{4}$$

$$\lambda_i = f_2(t, t_{0.5}) \tag{5}$$

where $\lambda$ is the fractional retained strain; $\epsilon^*$, expressed as a Taylor series expansion of $\lambda_{i-1}\epsilon_{i-1}$, is the retained strain; $t$ is the interpass time; $t_{0.5}$ is the 50 % recrystallisation time; $q(T_{i-1}, T_i)$ is a prescribed nonlinear function of $T_{i-1}$ and $T_i$; $f_1(.)$ and $f_2(.)$ are pre-specified nonlinear functions; $i$, the roll pass number; $j, h, g$ are the model coefficients; $T_{upper}$ is an experimentally determined temperature at which the material undergoes a permanent change in structure; and $T_{lower}$ is a temperature below which the material does not exhibit any plastic behaviour.

Model coefficients $a, b, c, d, f, g, h, j$ are either estimated in a batch mode (i.e., all the past data are assumed to be available simultaneously) or adapted recursively on-line (i.e., only a limited number of the past data is available) using a Kalman filter algorithm in order to provide the best model predictions [McFarlane, Telford, Petersen, 1991].

It is noted that these models are motivated by the desire to fit a nonlinear model of a special type, i.e., one which has an inverse hyperbolic sine function. But, since the basic operation is data fitting, i.e., to fit a model to the set of given data, it is possible to consider more general nonlinear models. These models may not have any ready interpretation in metallurgical terms, but these models may be better in fitting a nonlinear model to the given data set in the sense that it may give a better prediction of the output.

It has been shown (see, e.g., Hornik et al, 1989) that a class of artificial neural networks, viz., a multilayer perceptron with a single hidden layer can approximate any arbitrary input output function to an arbitrary degree of accuracy. Thus it is reasonable to experiment with different classes of artificial neural network or induction tree structures for fitting the set of given data and to examine which structure gives the best performance.

The structure of the paper is as follows: in section 2, a brief review of a special class of neural networks is given. In section 3, results in applying the neural network model to the plate mill data are given.

## 2    A Tree Based Neural Network model

Friedman [1991] introduced a new class of neural network architecture which is called MARS (Multivariate Adaptive Regression Spline). This class of methods can be interpreted as a tree of neurons, in which each leaf of the tree consists of a neuron. The model of the neuron may be a piecewise linear polynomial, or a cubic polynomial, with the knot as a variable. In view of the lack of space, we will refer the interested readers to Friedman's paper [1991] for details on this method.

## 3    Results

MARS has been applied to the platemill data. We have used the data in the following manner.

We concatenate different runs of the plate mill into a single time series. This consists of 2877 points corresponding to 180 individual plates with approximately 16 passes on each plate. There are 4 independent variables, viz., interpass time, temperature, strain, and strain rate. The desired output variable is the yield stress.

A plot of the individual variables, viz temperature, strain, strain rate, interpass time and stress versus time reveal that the variables can vary rather considerably over the entire time series. In addition, a plot of stress versus temperature, stress versus strain, stress versus strain rate and stress versus interpass time reveals that the functional dependence could be highly nonlinear.

We have chosen to use an additive model (Friedman [1991]), instead of the more general multivariate model, as this will allow us to observe any possible nonlinear functional dependencies of the output as a function of the inputs.

$$k_m = k_1 f_1(T) + k_2 f_2(\epsilon) + k_3 f_3(\dot{\epsilon}) + k_4 f_4(t) \qquad (6)$$

where $k_i, i = 1, 2, 3, 4$ are gains, and $f_i, i = 1, 2, 3, 4$ are piecewise nonlinear polynomial models found by MARS.

The results are as follows:

Both the piecewise linear polynomial and the piecewise cubic polynomial are used to study this set of data. It is found that the cubic polynomial gives a better fit than the linear polynomial fit. Figure 1(a) shows the error plot between the estimated output from a cubic spline fit, and the training data. It is observed that the error is very small. The maximum error is about -0.07. Figure 1(b) shows the plot of the predicted yield stress and the original yield stress over the set of training data.

These figures indicate that the cubic polynomial fit has captured most of the variation of the data. It is interesting to note that in this model, the interpass time

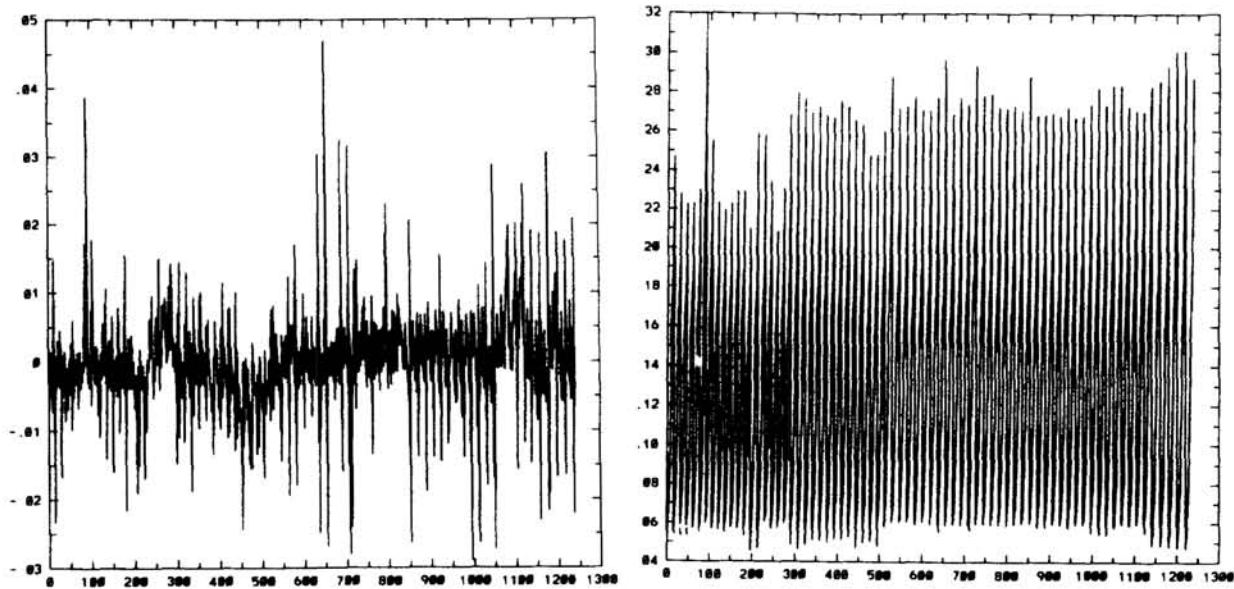

Figure 1: (a) The prediction error on the training data set (b) The prediction and the training data set superimposed

plays no significant part. This feature may be a peculiar aspect of this set of data points. It is not true in general.

It is found that the strain rate has the most influence on the data, followed by temperature, and followed by strain. The model, once obtained, can be used to predict the yield stress from a given set of temperature, strain, and strain rate.

Figure 2(a) shows the prediction error between the yield stress and the predicted yield stress on a set of testing data, i.e. the data which is not used to train the model and Figure 2(b) shows a plot of the predicted value of yield stress superimposed on the original yield stress.

It is observed that the prediction on the set of testing data is reasonable. This indicates that the MARS model has captured most of the dynamics underlying the original training data, and is capable of extending this captured knowledge onto a set of hitherto unseen data.

## 4    Comparison with the results obtained by conventional approaches

In order to compare the artificial neural network approach to more conventional methods for model tuning, the same data set was processed using:

1. A MARS model with cubic polynomials

2. An inverse hyperbolic sine law model using least square batch parameter tuning

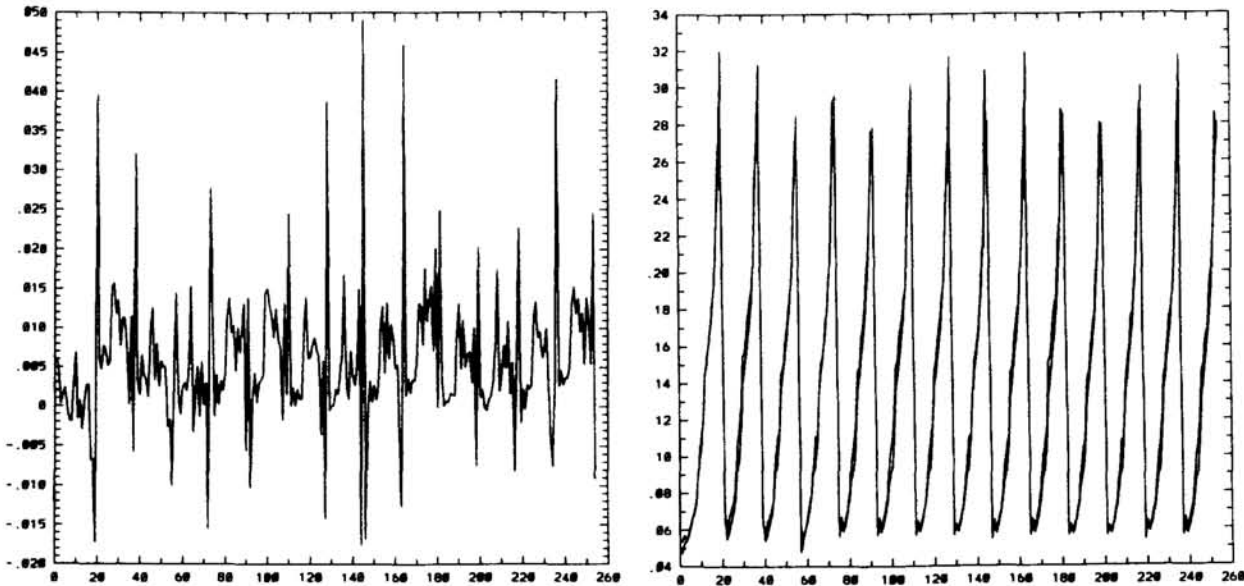

Figure 2: (a) The prediction error on the testing data set (b) The prediction and the testing data set superimposed

3. An inverse hyperbolic sine law model using a recursive least squares tuning

4. CART based classification [Briemen et. al., 1984]

5. C4.5 based method [Quinlan, 1986,1987]

6. Bayesian classification [Buntine, [1990]

In each case, we used a training data set of 78 plates (1242 passes) and a testing data set of 16 plates (252 passes). In the cases of CART, C4.5, and Bayesian classification methods, the yield stress variable is divided equally into 10 classes, and this is used as the desired output instead of the original real values.

The comparison of the results between MARS and the Kalman filter based approach are shown in the following table

|          | $B_{11}$ | $B_{12}$ | $A_{11}$ | $A_{12}$ | $C_{11}$ | $C_{12}$ |
|----------|------|------|------|------|------|------|
| mean%    | -.64 | 1.69 | -.64 | 2.38 | -0.2 | 4.5  |
| mean abs%| 4.61 | 4.22 | 4.61 | 5.3  | 3.5  | 5.3  |
| std %    | 6.26 | 5.11 | 6.26 | 6.25 | 4.7  | 4.9  |

where

$B_{11}$ = Batch Tuning: tuning model ( forgetting factors =1 in adaption) on the training data

$B_{12}$ = Batch Tuning: running tuned model on the testing data

$A_{11}$ = Adaptation: on the training data

$A_{12}$ = Adaptation: on the testing data

$C_{11}$ = MARS on the training data
$C_{12}$ = MARS on the testing data,

and $mean\% = mean((k_{meas} - k_{pred})/k_{meas})$,
$meanabs\% = mean(abs((k_{meas} - k_{pred})/k_{meas}))$,
$std\% = stdev((k_{meas} - k_{pred})/k_{meas})$; where mean and stdev stands for the mean and the standard deviations respectively, and $k_{meas}$, $k_{pred}$ represents the measured and predicted values of the yield stress respectively.

It is found that the MARS based model performs extremely well compared with the other methods. The standard deviation of the prediction errors in a MARS model is considerably less than the corresponding standard deviation of prediction errors in a Kalman filter type batch or online tuning model on the testing data set.

We have also compared MARS with both the CART based method and the C4.5 based method. As both CART and C4.5 operate only on an output category, rather than a continuous output value, it is necessary to convert the yield stress into a category type of variable. We have chosen to divide equally the yield stress into 10 classes. With this modification, the CART and C4.5 methods are readily applicable.

The following table summarises the results of this comparison. The values given are the percentage of the prediction error on the testing data set for various methods. In the case of MARS, we have converted the prediction error from a continuous variable into the corresponding classes as used in the CART and C4.5 methods.

| Bayes | CART | C4.5 | MARS |
|-------|-------|-------|------|
| 65.4 | 12.99 | 16.14 | 6.2 |

It is found that the MARS model is more consistent in predicting the output classes than either the CART method, the C4.5 based method, or the Bayesian classifier. The fact that the MARS model performs better than the CART model can be seen as a confirmation that the MARS model is a generalisation of the CART model (see Friedman [1991]). But it is rather surprising to see that the MARS model outperforms a Bayesian classifier.

The results are similar over a number of other typical data sets, e.g., when the interpass time variable becomes significant.

## 5   Conclusion

It is found that MARS can be applied to model the platemill data with very good accuracy. In terms of predictive power on unseen data, it performs better than the more traditional methods, e.g., Kalman filter batch or online tuning methods, CART, C4.5 or Bayesian classifier.

It is almost impossible to convert the MARS model into one given in section 1. The Hodgson-Collinson model places a breakpoint at a temperature of $925 \deg C$, while in the MARS model, the temperature breakpoints are found to be at $1017 \deg C$ and $1129 \deg C$ respectively. Hence it is difficult to convert the MARS model into those given by the Hodgson-Collinson model, the Kalman filter type models or vice

versa.

A possible improvement to the current MARS technique would be to restrict the breakpoints, so that they must exist within a temperature region where microstructural changes are known to occur.

# 6    Acknowledgement

The author acknowledges the assistance given by the staff at the BHP Melbourne Research Laboratory in providing the data, as well as in providing the background material in this paper. He specially thanks Dr D McFarlane in giving his generous time in assisting in the understanding of the more traditional approaches, and also for providing the results on the Kalman filtering approach. Also, he is indebted to Dr W Buntine, RIACS, NASA, Ames Research Center for providing an early version of the induction tree based programs.

# 7    References

Anderson, B.D.O., Moore, J.B., (1980). *Optimal Fitering*. Prentice Hall, Eaglewood, NJ.

Brieman, L., Friedman, J., Olshen, R.A., Stone, C.J., (1984). *Classification and Regrression Trees*. Wadworth, Belmont, CA.

Buntine, W, (1990). *A Theory of Learning Classification Rules*. PhD Thesis submitted to the University of Technology, Sydney.

Friedman, J, (1991). "Multivariate Adaptive Regression Splines". *Ann Stat.* to appear. ( Also, the implication of the paper on neural network models was presented orally in the 1990 NIPS Conference)

Hodgson, Collinson, (1990). Manuscript under preparation (authors are with BHP Research Lab., Melbourne, Australia).

Horihata, M, Motomura, M, (1988). "Theoretical analysis of 3-roll Rolling Process by the energy method". *Trans of the Iron and Steel Institute of Japan*, 28:6, 434-439.

Hornik, K., Stinchcombe, M., White, H., (1989). "Multilayer Feedforward Networks are Universal Approximators". *Neural Networks*, 2, 359-366.

McFarlane, D, Telford, A, Petersen, I, (1991). Manuscript under preparation

Quinlan, R. (1986). "Induction of Decision Trees". *Machine Learning*. 1, 81-106.

Quinlan, R. (1987). "Simplifying Decision Trees". *International J Man-Machine Studies*. 27, 221-234.

Underwood, L R, (1950). *The Rolling of Metals*. Chapman & Hall, London.